# A Kernel Statistical Test of Independence

**Arthur Gretton**
MPI for Biological Cybernetics
Tübingen, Germany
*arthur@tuebingen.mpg.de*

**Kenji Fukumizu**
Inst. of Statistical Mathematics
Tokyo Japan
*fukumizu@ism.ac.jp*

**Choon Hui Teo**
NICTA, ANU
Canberra, Australia
*choonhui.teo@gmail.com*

**Le Song**
NICTA, ANU
and University of Sydney
*lesong@it.usyd.edu.au*

**Bernhard Schölkopf**
MPI for Biological Cybernetics
Tübingen, Germany
*bs@tuebingen.mpg.de*

**Alexander J. Smola**
NICTA, ANU
Canberra, Australia
*alex.smola@gmail.com*

## Abstract

Although kernel measures of independence have been widely applied in machine learning (notably in kernel ICA), there is as yet no method to determine whether they have detected statistically significant dependence. We provide a novel test of the independence hypothesis for one particular kernel independence measure, the Hilbert-Schmidt independence criterion (HSIC). The resulting test costs $O(m^2)$, where $m$ is the sample size. We demonstrate that this test outperforms established contingency table and functional correlation-based tests, and that this advantage is greater for multivariate data. Finally, we show the HSIC test also applies to text (and to structured data more generally), for which no other independence test presently exists.

## 1 Introduction

Kernel independence measures have been widely applied in recent machine learning literature, most commonly in independent component analysis (ICA) [2, 11], but also in fitting graphical models [1] and in feature selection [22]. One reason for their success is that these criteria have a zero expected value if and only if the associated random variables are independent, when the kernels are universal (in the sense of [23]). There is presently no way to tell whether the *empirical estimates* of these dependence measures indicate a *statistically significant* dependence, however. In other words, we are interested in the threshold an empirical kernel dependence estimate must exceed, before we can dismiss with high probability the hypothesis that the underlying variables are independent.

Statistical tests of independence have been associated with a broad variety of dependence measures. Classical tests such as Spearman's $\rho$ and Kendall's $\tau$ are widely applied, however they are not guaranteed to detect all modes of dependence between the random variables. Contingency table-based methods, and in particular the power-divergence family of test statistics [17], are the best known general purpose tests of independence, but are limited to relatively low dimensions, since they require a partitioning of the space in which each random variable resides. Characteristic function-based tests [6, 13] have also been proposed, which are more general than kernel density-based tests [19], although to our knowledge they have been used only to compare univariate random variables.

In this paper we present three main results: first, and most importantly, we show how to test whether statistically significant dependence is detected by a particular kernel independence measure, the Hilbert Schmidt independence criterion (HSIC, from [9]). That is, we provide a fast ($O(m^2)$ for sample size $m$) and accurate means of obtaining a *threshold* which HSIC will only exceed with small probability, when the underlying variables are independent. Second, we show the distribution

of our empirical test statistic in the large sample limit can be straightforwardly parameterised in terms of kernels on the data. Third, we apply our test to structured data (in this case, by establishing the statistical dependence between a text and its translation). To our knowledge, ours is the first independence test for structured data.

We begin our presentation in Section 2, with a short overview of cross-covariance operators between RKHSs and their Hilbert-Schmidt norms: the latter are used to define the Hilbert Schmidt Independence Criterion (HSIC). In Section 3, we describe how to determine whether the dependence returned via HSIC is statistically significant, by proposing a hypothesis test with HSIC as its statistic. In particular, we show that this test can be parameterised using a combination of covariance operator norms and norms of mean elements of the random variables in feature space. Finally, in Section 4, we give our experimental results, both for testing dependence between random vectors (which could be used for instance to verify convergence in independent subspace analysis [25]), and for testing dependence between text and its translation. Software to implement the test may be downloaded from $\text{http}://\text{www.kyb.mpg.de/bs/people/arthur/indep.htm}$

## 2 Definitions and description of HSIC

Our problem setting is as follows:

**Problem 1** *Let $\mathbf{P}_{xy}$ be a Borel probability measure defined on a domain $\mathcal{X} \times \mathcal{Y}$, and let $\mathbf{P}_x$ and $\mathbf{P}_y$ be the respective marginal distributions on $\mathcal{X}$ and $\mathcal{Y}$. Given an i.i.d sample $Z := (X, Y) = \{(x_1, y_1), \ldots, (x_m, y_m)\}$ of size $m$ drawn independently and identically distributed according to $\mathbf{P}_{xy}$, does $\mathbf{P}_{xy}$ factorise as $\mathbf{P}_x \mathbf{P}_y$ (equivalently, we may write $x \perp\!\!\!\perp y$)?*

We begin with a description of our kernel dependence criterion, leaving to the following section the question of whether this dependence is significant. This presentation is largely a review of material from [9, 11, 22], the main difference being that we establish links to the characteristic function-based independence criteria in [6, 13]. Let $\mathcal{F}$ be an RKHS, with the continuous feature mapping $\phi(x) \in \mathcal{F}$ from each $x \in \mathcal{X}$, such that the inner product between the features is given by the kernel function $k(x, x') := \langle \phi(x), \phi(x') \rangle$. Likewise, let $\mathcal{G}$ be a second RKHS on $\mathcal{Y}$ with kernel $l(\cdot, \cdot)$ and feature map $\psi(y)$. Following [7], the cross-covariance operator $C_{xy} : \mathcal{G} \to \mathcal{F}$ is defined such that for all $f \in \mathcal{F}$ and $g \in \mathcal{G}$,

$$\langle f, C_{xy} g \rangle_{\mathcal{F}} = \mathbf{E}_{xy} \left( [f(x) - \mathbf{E}_x(f(x))] [g(y) - \mathbf{E}_y(g(y))] \right).$$

The cross-covariance operator itself can then be written

$$C_{xy} := \mathbf{E}_{xy}[(\phi(x) - \mu_x) \otimes (\psi(y) - \mu_y)], \tag{1}$$

where $\mu_x := \mathbf{E}_x \phi(x)$, $\mu_y := \mathbf{E}_y \phi(y)$, and $\otimes$ is the tensor product [9, Eq. 6]: this is a generalisation of the cross-covariance matrix between random vectors. When $\mathcal{F}$ and $\mathcal{G}$ are universal reproducing kernel Hilbert spaces (that is, dense in the space of bounded continuous functions [23]) on the compact domains $\mathcal{X}$ and $\mathcal{Y}$, then the largest singular value of this operator, $\|C_{xy}\|$, is zero if and only if $x \perp\!\!\!\perp y$ [11, Theorem 6]: the operator therefore induces an independence criterion, and can be used to solve Problem 1. The maximum singular value gives a criterion similar to that originally proposed in [18], but with more restrictive function classes (rather than functions of bounded variance). Rather than the maximum singular value, we may use the squared Hilbert-Schmidt norm (the sum of the squared singular values), which has a population expression

$$\text{HSIC}(\mathbf{P}_{xy}, \mathcal{F}, \mathcal{G}) = \mathbf{E}_{xx'yy'}[k(x, x')l(y, y')] + \mathbf{E}_{xx'}[k(x, x')]\mathbf{E}_{yy'}[l(y, y')]$$
$$- 2\mathbf{E}_{xy} \left[ \mathbf{E}_{x'}[k(x, x')]\mathbf{E}_{y'}[l(y, y')]] \right] \tag{2}$$

(assuming the expectations exist), where $x'$ denotes an independent copy of $x$ [9, Lemma 1]: we call this the Hilbert-Schmidt independence criterion (HSIC).

We now address the problem of estimating $\text{HSIC}(\mathbf{P}_{xy}, \mathcal{F}, \mathcal{G})$ on the basis of the sample $Z$. An unbiased estimator of (2) is a sum of three U-statistics [21, 22],

$$\text{HSIC}(Z) = \frac{1}{(m)_2} \sum_{(i,j) \in \mathbf{i}_2^m} k_{ij} l_{ij} + \frac{1}{(m)_4} \sum_{(i,j,q,r) \in \mathbf{i}_4^m} k_{ij} l_{qr} - 2\frac{1}{(m)_3} \sum_{(i,j,q) \in \mathbf{i}_3^m} k_{ij} l_{iq}, \tag{3}$$

where $(m)_n := \frac{m!}{(m-n)!}$, the index set $\mathbf{i}_r^m$ denotes the set all $r$-tuples drawn without replacement from the set $\{1, \ldots, m\}$, $k_{ij} := k(x_i, x_j)$, and $l_{ij} := l(y_i, y_j)$. For the purpose of testing independence, however, we will find it easier to use an alternative, biased empirical estimate [9, Definition 2], obtained by replacing the U-statistics with V-statistics[1]

$$\text{HSIC}_b(Z) = \frac{1}{m^2} \sum_{i,j}^m k_{ij} l_{ij} + \frac{1}{m^4} \sum_{i,j,q,r}^m k_{ij} l_{qr} - 2\frac{1}{m^3} \sum_{i,j,q}^m k_{ij} l_{iq} = \frac{1}{m^2} \text{trace}(\mathbf{KHLH}), \quad (4)$$

where the summation indices now denote all $r$-tuples drawn with replacement from $\{1, \ldots, m\}$ ($r$ being the number of indices below the sum), $\mathbf{K}$ is the $m \times m$ matrix with entries $k_{ij}$, $\mathbf{H} = \mathbf{I} - \frac{1}{m}\mathbf{1}\mathbf{1}^\top$, and $\mathbf{1}$ is an $m \times 1$ vector of ones (the cost of computing this statistic is $\text{O}(m^2)$). When a Gaussian kernel $k_{ij} := \exp\left(-\sigma^{-2} \|x_i - x_j\|^2\right)$ is used (or a kernel deriving from [6, Eq. 4.10]), the latter statistic is equivalent to the characteristic function-based statistic [6, Eq. 4.11] and the $T2_n$ statistic of [13, p. 54]: details are reproduced in [10] for comparison. Our setting allows for more general kernels, however, such as kernels on strings (as in our experiments in Section 4) and graphs (see [20] for further details of kernels on structures): this is not possible under the characteristic function framework, which is restricted to Euclidean spaces ($\mathbb{R}^d$ in the case of [6, 13]). As pointed out in [6, Section 5], the statistic in (4) can also be linked to the original quadratic test of Rosenblatt [19] given an appropriate kernel choice; the main differences being that characteristic function-based tests (and RKHS-based tests) are not restricted to using kernel densities, nor should they reduce their kernel width with increasing sample size. Another related test described in [4] is based on the functional canonical correlation between $\mathcal{F}$ and $\mathcal{G}$, rather than the covariance: in this sense the test statistic resembles those in [2]. The approach in [4] differs with both the present work and [2], however, in that the function spaces $\mathcal{F}$ and $\mathcal{G}$ are represented by finite sets of basis functions (specifically B-spline kernels) when computing the empirical test statistic.

## 3 Test description

We now describe a statistical test of independence for two random variables, based on the test statistic $\text{HSIC}_b(Z)$. We begin with a more formal introduction to the framework and terminology of statistical hypothesis testing. Given the i.i.d. sample $Z$ defined earlier, the statistical test, $\mathcal{T}(Z) : (\mathcal{X} \times \mathcal{Y})^m \mapsto \{0, 1\}$ is used to distinguish between the null hypothesis $\mathcal{H}_0 : \mathbf{P}_{xy} = \mathbf{P}_x\mathbf{P}_y$ and the alternative hypothesis $\mathcal{H}_1 : \mathbf{P}_{xy} \neq \mathbf{P}_x\mathbf{P}_y$. This is achieved by comparing the test statistic, in our case $\text{HSIC}_b(Z)$, with a particular threshold: if the threshold is exceeded, then the test rejects the null hypothesis (bearing in mind that a zero population HSIC indicates $\mathbf{P}_{xy} = \mathbf{P}_x\mathbf{P}_y$). The acceptance region of the test is thus defined as any real number below the threshold. Since the test is based on a finite sample, it is possible that an incorrect answer will be returned: the Type I error is defined as the probability of rejecting $\mathcal{H}_0$ based on the observed sample, despite $x$ and $y$ being independent. Conversely, the Type II error is the probability of accepting $\mathbf{P}_{xy} = \mathbf{P}_x\mathbf{P}_y$ when the underlying variables are dependent. The level $\alpha$ of a test is an upper bound on the Type I error, and is a design parameter of the test, used to set the test threshold. A consistent test achieves a level $\alpha$, and a Type II error of zero, in the large sample limit.

How, then, do we set the threshold of the test given $\alpha$? The approach we adopt here is to derive the asymptotic distribution of the empirical estimate $\text{HSIC}_b(Z)$ of $\text{HSIC}(\mathbf{P}_{xy}, \mathcal{F}, \mathcal{G})$ under $\mathcal{H}_0$. We then use the $1 - \alpha$ quantile of this distribution as the test threshold.[2] Our presentation in this section is therefore divided into two parts. First, we obtain the distribution of $\text{HSIC}_b(Z)$ under both $\mathcal{H}_0$ and $\mathcal{H}_1$; the latter distribution is also needed to ensure consistency of the test. We shall see, however, that the null distribution has a complex form, and cannot be evaluated directly. Thus, in the second part of this section, we describe ways to accurately approximate the $1 - \alpha$ quantile of this distribution.

**Asymptotic distribution of** $\text{HSIC}_b(Z)$  We now describe the distribution of the test statistic in (4) The first theorem holds under $\mathcal{H}_1$.

**Theorem 1** *Let*

$$h_{ijqr} = \frac{1}{4!} \sum_{(t,u,v,w)}^{(i,j,q,r)} k_{tu}l_{tu} + k_{tu}l_{vw} - 2k_{tu}l_{tv}, \tag{5}$$

*where the sum represents all ordered quadruples $(t, u, v, w)$ drawn without replacement from $(i, j, q, r)$, and assume $\mathbf{E}\left(h^2\right) < \infty$. Under $\mathcal{H}_1$, $\mathrm{HSIC}_b(Z)$ converges in distribution as $m \to \infty$ to a Gaussian according to*

$$m^{\frac{1}{2}} \left(\mathrm{HSIC}_b(Z) - \mathrm{HSIC}(\mathbf{P}_{xy}, \mathcal{F}, \mathcal{G})\right) \xrightarrow{D} \mathcal{N}\left(0, \sigma_u^2\right). \tag{6}$$

*The variance is $\sigma_u^2 = 16\left(\mathbf{E}_i\left(\mathbf{E}_{j,q,r}h_{ijqr}\right)^2 - \mathrm{HSIC}(\mathbf{P}_{xy}, \mathcal{F}, \mathcal{G})\right)$, where $\mathbf{E}_{j,q,r} := \mathbf{E}_{z_j, z_q, z_r}$.*

**Proof** We first rewrite (4) as a single V-statistic,

$$\mathrm{HSIC}_b(Z) = \frac{1}{m^4} \sum_{i,j,q,r}^{m} h_{ijqr}, \tag{7}$$

where we note that $h_{ijqr}$ defined in (5) does not change with permutation of its indices. The associated U-statistic $\mathrm{HSIC}_s(Z)$ converges in distribution as (6) with variance $\sigma_u^2$ [21, Theorem 5.5.1(A)]: see [22]. Since the difference between $\mathrm{HSIC}_b(Z)$ and $\mathrm{HSIC}_s(Z)$ drops as $1/m$ (see [9], or Theorem 3 below), $\mathrm{HSIC}_b(Z)$ converges asymptotically to the same distribution. ∎

The second theorem applies under $\mathcal{H}_0$

**Theorem 2** *Under $\mathcal{H}_0$, the U-statistic $\mathrm{HSIC}_s(Z)$ corresponding to the V-statistic in (7) is degenerate, meaning $\mathbf{E}_i h_{ijqr} = 0$. In this case, $\mathrm{HSIC}_b(Z)$ converges in distribution according to [21, Section 5.5.2]*

$$m\mathrm{HSIC}_b(Z) \xrightarrow{D} \sum_{l=1}^{\infty} \lambda_l z_l^2, \tag{8}$$

*where $z_l \sim \mathcal{N}(0, 1)$ i.i.d., and $\lambda_l$ are the solutions to the eigenvalue problem*

$$\lambda_l \psi_l(z_j) = \int h_{ijqr} \psi_l(z_i) dF_{i,q,r},$$

*where the integral is over the distribution of variables $z_i$, $z_q$, and $z_r$.*

**Proof** This follows from the discussion of [21, Section 5.5.2], making appropriate allowance for the fact that we are dealing with a V-statistic (which is why the terms in (8) are not centred: in the case of a U-statistic, the sum would be over terms $\lambda_l(z_l^2 - 1)$). ∎

**Approximating the $1 - \alpha$ quantile of the null distribution** A hypothesis test using $\mathrm{HSIC}_b(Z)$ could be derived from Theorem 2 above by computing the $(1 - \alpha)$th quantile of the distribution (8), where consistency of the test (that is, the convergence to zero of the Type II error for $m \to \infty$) is guaranteed by the decay as $m^{-1}$ of the variance of $\mathrm{HSIC}_b(Z)$ under $\mathcal{H}_1$. The distribution under $\mathcal{H}_0$ is complex, however: the question then becomes how to accurately approximate its quantiles.

One approach, taken by [6], is to use a Monte Carlo resampling technique: the ordering of the $Y$ sample is permuted repeatedly while that of $X$ is kept fixed, and the $1 - \alpha$ quantile is obtained from the resulting distribution of $\mathrm{HSIC}_b$ values. This can be very expensive, however. A second approach, suggested in [13, p. 34], is to approximate the null distribution as a two-parameter Gamma distribution [12, p. 343, p. 359]: this is one of the more straightforward approximations of an infinite sum of $\chi^2$ variables (see [12, Chapter 18.8] for further ways to approximate such distributions; in particular, we wish to avoid using moments of order greater than two, since these can become expensive to compute). Specifically, we make the approximation

$$m\mathrm{HSIC}_b(Z) \sim \frac{x^{\alpha-1}e^{-x/\beta}}{\beta^\alpha \Gamma(\alpha)} \quad \text{where} \quad \alpha = \frac{(\mathbf{E}(\mathrm{HSIC}_b(Z)))^2}{\mathrm{var}(\mathrm{HSIC}_b(Z))}, \quad \beta = \frac{m\mathrm{var}(\mathrm{HSIC}_b(Z))}{\mathbf{E}(\mathrm{HSIC}_b(Z))}. \tag{9}$$

An illustration of the cumulative distribution function (CDF) obtained via the Gamma approximation is given in Figure 1, along with an empirical CDF obtained by repeated draws of $\text{HSIC}_b$. We note the Gamma approximation is quite accurate, especially in areas of high probability (which we use to compute the test quantile). The accuracy of this approximation will be further evaluated experimentally in Section 4.

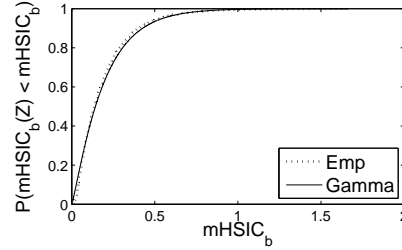

Figure 1: $m\text{HSIC}_b$ cumulative distribution function (*Emp*) under $\mathcal{H}_0$ for $m = 200$, obtained empirically using 5000 independent draws of $m\text{HSIC}_b$. The two-parameter Gamma distribution (*Gamma*) is fit using $\alpha = 1.17$ and $\beta = 8.3 \times 10^{-4}$ in (9), with mean and variance computed via Theorems 3 and 4.

To obtain the Gamma distribution from our observations, we need empirical estimates for $\mathbf{E}(\text{HSIC}_b(Z))$ and $\text{var}(\text{HSIC}_b(Z))$ under the null hypothesis. Expressions for these quantities are given in [13, pp. 26-27], however these are in terms of the joint and marginal characteristic functions, and not in our more general kernel setting (see also [14, p. 313]). In the following two theorems, we provide much simpler expressions for both quantities, in terms of norms of mean elements $\mu_x$ and $\mu_y$, and the covariance operators

$$C_{xx} := \mathbf{E}_x[(\phi(x) - \mu_x) \otimes (\phi(x) - \mu_x)]$$

and $C_{yy}$, in feature space. The main advantage of our new expressions is that they are computed entirely in terms of kernels, which makes possible the application of the test to any domains on which kernels can be defined, and not only $\mathbb{R}^d$.

**Theorem 3** *Under $\mathcal{H}_0$,*

$$\mathbf{E}(\text{HSIC}_b(Z)) = \frac{1}{m}\text{Tr}C_{xx}\text{Tr}C_{yy} = \frac{1}{m}\left(1 + \|\mu_x\|^2\|\mu_y\|^2 - \|\mu_x\|^2 - \|\mu_y\|^2\right), \quad (10)$$

*where the second equality assumes $k_{ii} = l_{ii} = 1$. An empirical estimate of this statistic is obtained by replacing the norms above with $\widehat{\|\mu_x\|^2} = (m)_2^{-1}\sum_{(i,j)\in\mathbf{i}_2^m} k_{ij}$, bearing in mind that this results in a (generally negligible) bias of $\text{O}(m^{-1})$ in the estimate of $\|\mu_x\|^2\|\mu_y\|^2$.*

**Theorem 4** *Under $\mathcal{H}_0$,*

$$\text{var}(\text{HSIC}_b(Z)) = \frac{2(m-4)(m-5)}{(m)_4}\|C_{xx}\|_{\text{HS}}^2\|C_{yy}\|_{\text{HS}}^2 + \text{O}(m^{-3}).$$

*Denoting by $\odot$ the entrywise matrix product, $A^{.2}$ the entrywise matrix power, and $\mathbf{B} = ((\mathbf{HKH}) \odot (\mathbf{HLH}))^{.2}$, an empirical estimate with negligible bias may be found by replacing the product of covariance operator norms with $\mathbf{1}^\top(\mathbf{B} - \text{diag}(\mathbf{B}))\mathbf{1}$: this is slightly more efficient than taking the product of the empirical operator norms (although the scaling with $m$ is unchanged).*

Proofs of both theorems may be found in [10], where we also compare with the original characteristic function-based expressions in [13]. We remark that these parameters, like the original test statistic in (4), may be computed in $\text{O}(m^2)$.

## 4 Experiments

General tests of statistical independence are most useful for data having complex interactions that simple correlation does not detect. We investigate two cases where this situation arises: first, we test vectors in $\mathbb{R}^d$ which have a dependence relation but no correlation, as occurs in independent subspace analysis; and second, we study the statistical dependence between a text and its translation.

**Independence of subspaces** One area where independence tests have been applied is in determining the convergence of algorithms for independent component analysis (ICA), which involves separating random variables that have been linearly mixed, using only their mutual independence. ICA generally entails optimisation over a non-convex function (including when HSIC is itself the optimisation criterion [9]), and is susceptible to local minima, hence the need for these tests (in fact, for classical approaches to ICA, the *global* minimum of the optimisation might not correspond to independence for certain source distributions). Contingency table-based tests have been applied [15]

in this context, while the test of [13] has been used in [14] for verifying ICA outcomes when the data are stationary random processes (through using a subset of samples with a sufficiently large delay between them). Contingency table-based tests may be less useful in the case of independent subspace analysis (ISA, see e.g. [25] and its bibliography), where higher dimensional independent random vectors are to be separated. Thus, characteristic function-based tests [6, 13] and kernel independence measures might work better for this problem.

In our experiments, we tested the independence of random vectors, as a way of verifying the solutions of independent subspace analysis. We assumed for ease of presentation that our subspaces have respective dimension $d_x = d_y = d$, but this is not required. The data were constructed as follows. First, we generated $m$ samples of two univariate random variables, each drawn at random from the ICA benchmark densities in [11, Table 3]: these include super-Gaussian, sub-Gaussian, multimodal, and unimodal distributions. Second, we mixed these random variables using a rotation matrix parameterised by an angle $\theta$, varying from 0 to $\pi/4$ (a zero angle means the data are independent, while dependence becomes easier to detect as the angle increases to $\pi/4$: see the two plots in Figure 2, top left). Third, we appended $d - 1$ dimensional Gaussian noise of zero mean and unit standard deviation to each of the mixtures. Finally, we multiplied each resulting vector by an independent random $d$-dimensional orthogonal matrix, to obtain vectors dependent across all observed dimensions. We emphasise that classical approaches (such as Spearman's $\rho$ or Kendall's $\tau$) are completely unable to find this dependence, since the variables are uncorrelated; nor can we recover the subspace in which the variables are dependent using PCA, since this subspace has the same second order properties as the noise. We investigated sample sizes $m = 128, 512, 1024, 2048$, and $d = 1, 2, 4$.

We compared two different methods for computing the $1 - \alpha$ quantile of the HSIC null distribution: repeated random permutation of the $Y$ sample ordering as in [6] (*HSICp*), where we used 200 permutations; and Gamma approximation (*HSICg*) as in [13], based on (9). We used a Gaussian kernel, with kernel size set to the median distance between points in input space. We also compared with two alternative tests, the first based on a discretisation of the variables, and the second on functional canonical correlation. The discretisation based test was a power-divergence contingency table test from [17] (*PD*), which consisted in partitioning the space, counting the number of samples falling in each partition, and comparing this with the number of samples that would be expected under the null hypothesis (the test we used, described in [15], is more refined than this short description would suggest). Rather than a uniform space partitioning, we divided our space into roughly equiprobable bins as in [15], using a Gessaman partition for higher dimensions [5, Figure 21.4] (Ku and Fine did not specify a space partitioning strategy for higher dimensions, since they dealt only with univariate random variables). All remaining parameters were set according to [15]. The functional correlation-based test (*fCorr*) is described in [4]: the main differences with respect to our test are that it uses the spectrum of the functional correlation operator, rather than the covariance operator; and that it approximates the RKHSs $\mathcal{F}$ and $\mathcal{G}$ by finite sets of basis functions. Parameter settings were as in [4, Table 1], with the second order B-spline kernel and a twofold dyadic partitioning. Note that *fCorr* applies only in the univariate case. Results are plotted in Figure 2 (average over 500 repetitions). The $y$-intercept on these plots corresponds to the acceptance rate of $\mathcal{H}_0$ at independence, or $1 - (\text{Type I error})$, and should be close to the design parameter of $1 - \alpha = 0.95$. Elsewhere, the plots indicate acceptance of $\mathcal{H}_0$ where the underlying variables are dependent, i.e. the Type II error.

As expected, we observe that dependence becomes easier to detect as $\theta$ increases from 0 to $\pi/4$, when $m$ increases, and when $d$ decreases. The *PD* and *fCorr* tests perform poorly at $m = 128$, but approach the performance of HSIC-based tests for increasing $m$ (although *PD* remains slightly worse than *HSIC* at $m = 512$ and $d = 1$, while *fCorr* becomes slightly worse again than *PD*). *PD* also scales very badly with $d$, and never rejects the null hypothesis when $d = 4$, even for $m = 2048$. Although HSIC-based tests are unreliable for small $\theta$, they generally do well as $\theta$ approaches $\pi/4$ (besides $m = 128$, $d = 2$). We also emphasise that *HSICp* and *HSICg* perform identically, although *HSICp* is far more costly (by a factor of around 100, given the number of permutations used).

**Dependence and independence between text** In this section, we demonstrate independence testing on text. Our data are taken from the Canadian Hansard corpus ($\text{http}://\text{www.isi.edu/natural} - \text{language/download/hansard/}$). These consist of the official records of the 36th Canadian parliament, in English and French. We used debate transcripts on the three topics of Agriculture, Fisheries, and Immigration, due to the relatively large volume of data in these categories. Our goal was to test whether there exists a statistical dependence between

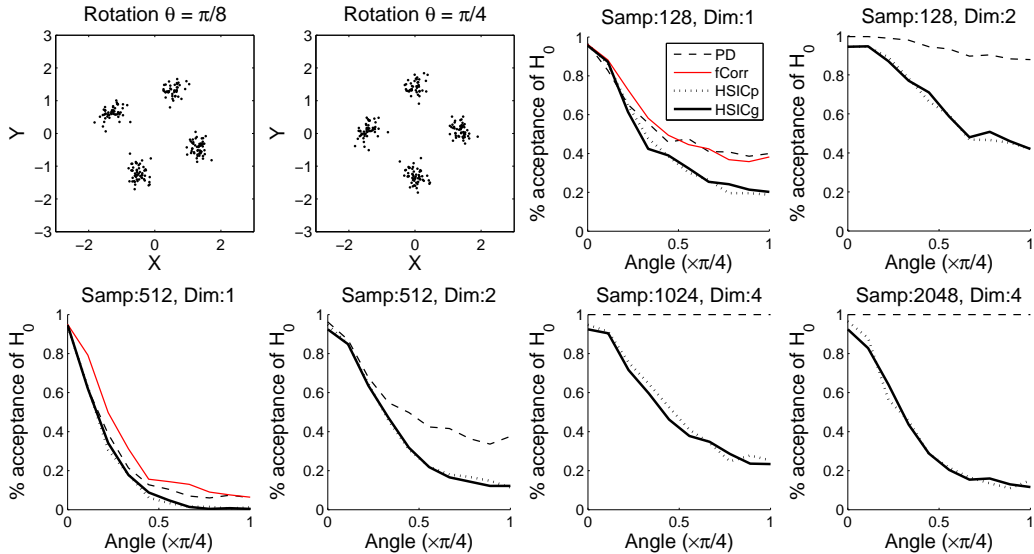

Figure 2: **Top left plots:** Example dataset for $d = 1$, $m = 200$, and rotation angles $\theta = \pi/8$ (left) and $\theta = \pi/4$ (right). In this case, both sources are mixtures of two Gaussians (source *(g)* in [11, Table 3]). We remark that the random variables appear "more dependent" as the angle $\theta$ increases, although their correlation is always zero. **Remaining plots:** Rate of acceptance of $\mathcal{H}_0$ for the *PD*, *fCorr*, *HSICp*, and *HSICg* tests. "Samp" is the number $m$ of samples, and "dim" is the dimension $d$ of $x$ and $y$.

English text and its French translation. Our dependent data consisted of a set of paragraph-long (5 line) English extracts and their French translations. For our independent data, the English paragraphs were matched to random French paragraphs on the same topic: for instance, an English paragraph on fisheries would always be matched with a French paragraph on fisheries. This was designed to prevent a simple vocabulary check from being used to tell when text was mismatched. We also ignored lines shorter than five words long, since these were not always part of the text (e.g. identification of the person speaking). We used the $k$-spectrum kernel of [16], computed according to the method of [24]. We set $k = 10$ for both languages, where this was chosen by cross validating on an SVM classifier for Fisheries vs National Defense, separately for each language (performance was not especially sensitive to choice of $k$; $k = 5$ also worked well). We compared this kernel with a simple kernel between bags of words [3, pp. 186–189]. Results are in Table 1.

Our results demonstrate the excellent performance of the *HSICp* test on this task: even for small sample sizes, *HSICp* with a spectral kernel always achieves zero Type II error, and a Type I error close to the design value (0.95). We further observe for $m = 10$ that *HSICp* with the spectral kernel always has better Type II error than the bag-of words kernel. This suggests that a kernel with a more sophisticated encoding of text structure induces a more sensitive test, although for larger sample sizes, the advantage vanishes. The *HSICg* test does less well on this data, always accepting $\mathcal{H}_0$ for $m = 10$, and returning a Type I error of zero, rather than the design value of 5%, when $m = 50$. It appears that this is due to a very low variance estimate returned by the Theorem 4 expression, which could be caused by the high diagonal dominance of kernels on strings. Thus, while the test threshold for *HSICg* at $m = 50$ still fell between the dependent and independent values of $\mathrm{HSIC}_b$, this was not the result of an accurate modelling of the null distribution. We would therefore recommend the permutation approach for this problem. Finally, we also tried testing with 2-line extracts and 10-line extracts, which yielded similar results.

## 5   Conclusion

We have introduced a test of whether significant statistical dependence is obtained by a kernel dependence measure, the Hilbert-Schmidt independence criterion (HSIC). Our test costs $\mathrm{O}(m^2)$ for sample size $m$. In our experiments, HSIC-based tests always outperformed the contingency table [17] and functional correlation [4] approaches, for both univariate random variables and higher dimensional vectors which were dependent but uncorrelated. We would therefore recommend HSIC-based tests for checking the convergence of independent component analysis and independent subspace analysis. Finally, our test also applies on structured domains, being able to detect the dependence

Table 1: Independence tests for cross-language dependence detection. Topics are in the first column, where the total number of 5-line extracts for each dataset is in parentheses. BOW(10) denotes a bag of words kernel and $m = 10$ sample size, Spec(50) is a $k$-spectrum kernel with $m = 50$. The first entry in each cell is the null acceptance rate of the test under $\mathcal{H}_0$ (i.e. $1 - (\text{Type I error})$; should be near 0.95); the second entry is the null acceptance rate under $\mathcal{H}_1$ (the Type II error, small is better). Each entry is an average over 300 repetitions.

| Topic | BOW(10) | | Spec(10) | | BOW(50) | | Spec(50) | |
|---|---|---|---|---|---|---|---|---|
| | HSICg | HSICp | HSICg | HSICp | HSICg | HSICp | HSICg | HSICp |
| Agriculture (555) | 1.00, 0.99 | 0.94, 0.18 | 1.00, 1.00 | 0.95, 0.00 | 1.00, 0.00 | 0.93, 0.00 | 1.00, 0.00 | 0.95, 0.00 |
| Fisheries (408) | 1.00, 1.00 | 0.94, 0.20 | 1.00, 1.00 | 0.94, 0.00 | 1.00, 0.00 | 0.93, 0.00 | 1.00, 0.00 | 0.95, 0.00 |
| Immigration (289) | 1.00, 1.00 | 0.96, 0.09 | 1.00, 1.00 | 0.91, 0.00 | 0.99, 0.00 | 0.94, 0.00 | 1.00, 0.00 | 0.95, 0.00 |

of passages of text and their translation.Another application along these lines might be in testing dependence between data of completely different types, such as images and captions.

**Acknowledgements:** NICTA is funded through the Australian Government's *Backing Australia's Ability* initiative, in part through the ARC. This work was supported in part by the IST Programme of the European Community, under the PASCAL Network of Excellence, IST-2002-506778.

## Footnotes

[1]The U- and V-statistics differ in that the latter allow indices of different sums to be equal.

[2]An alternative would be to use a large deviation bound, as provided for instance by [9] based on Hoeffding's inequality. It has been reported in [8], however, that such bounds are generally too loose for hypothesis testing.

# References

[1] F. Bach and M. Jordan. Tree-dependent component analysis. In *UAI 18*, 2002.

[2] F. R. Bach and M. I. Jordan. Kernel independent component analysis. *J. Mach. Learn. Res.*, 3:1–48, 2002.

[3] I. Calvino. *If on a winter's night a traveler*. Harvest Books, Florida, 1982.

[4] J. Dauxois and G. M. Nkiet. Nonlinear canonical analysis and independence tests. *Ann. Statist.*, 26(4):1254–1278, 1998.

[5] L. Devroye, L. Györfi, and G. Lugosi. *A Probabilistic Theory of Pattern Recognition*. Number 31 in Applications of mathematics. Springer, New York, 1996.

[6] Andrey Feuerverger. A consistent test for bivariate dependence. *International Statistical Review*, 61(3):419–433, 1993.

[7] K. Fukumizu, F. R. Bach, and M. I. Jordan. Dimensionality reduction for supervised learning with reproducing kernel Hilbert spaces. *Journal of Machine Learning Research*, 5:73–99, 2004.

[8] A. Gretton, K. Borgwardt, M. Rasch, B. Schölkopf, and A. Smola. A kernel method for the two-sample-problem. In *NIPS 19*, pages 513–520, Cambridge, MA, 2007. MIT Press.

[9] A. Gretton, O. Bousquet, A.J. Smola, and B. Schölkopf. Measuring statistical dependence with Hilbert-Schmidt norms. In *ALT*, pages 63–77, 2005.

[10] A. Gretton, K. Fukumizu, C.-H. Teo, L. Song, B. Schölkopf, and A. Smola. A kernel statistical test of independence. Technical Report 168, MPI for Biological Cybernetics, 2008.

[11] A. Gretton, R. Herbrich, A. Smola, O. Bousquet, and B. Schölkopf. Kernel methods for measuring independence. *J. Mach. Learn. Res.*, 6:2075–2129, 2005.

[12] N. L. Johnson, S. Kotz, and N. Balakrishnan. *Continuous Univariate Distributions. Volume 1 (Second Edition)*. John Wiley and Sons, 1994.

[13] A. Kankainen. *Consistent Testing of Total Independence Based on the Empirical Characteristic Function*. PhD thesis, University of Jyväskylä, 1995.

[14] Juha Karvanen. A resampling test for the total independence of stationary time series: Application to the performance evaluation of ica algorithms. *Neural Processing Letters*, 22(3):311 – 324, 2005.

[15] C.-J. Ku and T. Fine. Testing for stochastic independence: application to blind source separation. *IEEE Transactions on Signal Processing*, 53(5):1815–1826, 2005.

[16] C. Leslie, E. Eskin, and W. S. Noble. The spectrum kernel: A string kernel for SVM protein classification. In *Pacific Symposium on Biocomputing*, pages 564–575, 2002.

[17] T. Read and N. Cressie. *Goodness-Of-Fit Statistics for Discrete Multivariate Analysis*. Springer-Verlag, New York, 1988.

[18] A. Rényi. On measures of dependence. *Acta Math. Acad. Sci. Hungar.*, 10:441–451, 1959.

[19] M. Rosenblatt. A quadratic measure of deviation of two-dimensional density estimates and a test of independence. *The Annals of Statistics*, 3(1):1–14, 1975.

[20] B. Schölkopf, K. Tsuda, and J.-P. Vert. *Kernel Methods in Computational Biology*. MIT Press, 2004.

[21] R. Serfling. *Approximation Theorems of Mathematical Statistics*. Wiley, New York, 1980.

[22] L. Song, A. Smola, A. Gretton, K. Borgwardt, and J. Bedo. Supervised feature selection via dependence estimation. In *Proc. Intl. Conf. Machine Learning*, pages 823–830. Omnipress, 2007.

[23] I. Steinwart. The influence of the kernel on the consistency of support vector machines. *Journal of Machine Learning Research*, 2, 2002.

[24] C. H. Teo and S. V. N. Vishwanathan. Fast and space efficient string kernels using suffix arrays. In *ICML*, pages 929–936, 2006.

[25] F.J. Theis. Towards a general independent subspace analysis. In *NIPS 19*, 2007.

